# Optimal sub-graphical models

**Mukund Narasimhan**\* **and Jeff Bilmes**\*
Dept. of Electrical Engineering
University of Washington
Seattle, WA 98195
{mukundn,bilmes}@ee.washington.edu

## Abstract

We investigate the problem of reducing the complexity of a graphical model $(G, P_G)$ by finding a subgraph $H$ of $G$, chosen from a class of subgraphs $\mathcal{H}$, such that $H$ is optimal with respect to KL-divergence. We do this by first defining a decomposition tree representation for $G$, which is closely related to the junction-tree representation for $G$. We then give an algorithm which uses this representation to compute the optimal $H \in \mathcal{H}$. Gavril [2] and Tarjan [3] have used graph separation properties to solve several combinatorial optimization problems when the size of the minimal separators in the graph is bounded. We present an extension of this technique which applies to some important choices of $\mathcal{H}$ even when the size of the minimal separators of $G$ are arbitrarily large. In particular, this applies to problems such as finding an optimal subgraphical model over a $(k-1)$-tree of a graphical model over a $k$-tree (for arbitrary $k$) and selecting an optimal subgraphical model with (a constant) $d$ fewer edges with respect to KL-divergence can be solved in time polynomial in $|V(G)|$ using this formulation.

## 1 Introduction and Preliminaries

The complexity of inference in graphical models is typically exponential in some parameter of the graph, such as the size of the largest clique. Therefore, it is often required to find a subgraphical model that has lower complexity (smaller clique size) without introducing a large error in inference results. The KL-divergence between the original probability distribution and the probability distribution on the simplified graphical model is often used to measure the impact on inference. Existing techniques for reducing the complexity of graphical models including annihilation and edge-removal [4] are greedy in nature and cannot make any guarantees regarding the optimality of the solution. This problem is NP-complete [9] and so, in general, one cannot expect a polynomial time algorithm to find the optimal solution. However, we show that when we restrict the problem to some sets of subgraphs, the optimal solution can be found quite quickly using a dynamic programming algorithm in time polynomial in the tree-width of the graph.

### 1.1 Notation and Terminology

A graph $G = (V, E)$ is said to be triangulated if every cycle of length greater than 3 has a chord. A clique of $G$ is a non-empty set $S \subseteq V$ such that $\{a, b\} \in E$ for all

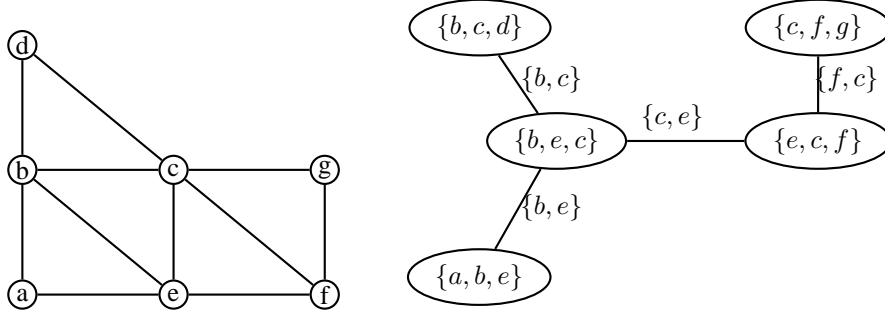

Figure 1: A triangulated graph $G$ and a junction-tree for $G$

$a, b \in S$. A clique $S$ is maximal if $S$ is not properly contained in another clique. If $\alpha$ and $\beta$ are non-adjacent vertices of $G$ then a set of vertices $S \subseteq V \setminus \{\alpha, \beta\}$ is called an $(\alpha, \beta)$-separator if $\alpha$ and $\beta$ are in distinct components of $G[V \setminus S]$. $S$ is a minimal $(\alpha, \beta)$-separator if no proper subset of $S$ is an $(\alpha, \beta)$-separator. $S$ is said to be a minimal separator if $S$ is a minimal $(\alpha, \beta)$-separator for some non adjacent $a, b \in V$. If $T = (\mathcal{K}, \mathcal{S})$ is a junction-tree for $G$ (see [7]), then the nodes $\mathcal{K}$ of $T$ correspond to the maximal-cliques of $G$, while the links $\mathcal{S}$ correspond to minimal separators of $G$ (We reserve the terms vertices/edges for elements of $G$, and nodes/links for the elements of $T$). If $G$ is triangulated, then the number of maximal cliques is at most $|V|$. For example, in the graph $G$ shown in Figure 1, $\mathcal{K} = \{\{b, c, d\}, \{a, b, e\}, \{b, e, c\}, \{e, c, f\}, \{c, f, g\}\}$. The links $\mathcal{S}$ of $T$ correspond to minimal-separators of $G$ in the following way. If $V_i V_j \in \mathcal{S}$ (where $V_i, V_j \in \mathcal{K}$ and hence are cliques of $G$), then $V_i \cap V_j \neq \phi$. We label each edge $V_i V_j \in \mathcal{S}$ with the set $V_{ij} = V_i \cap V_j$, which is a non-empty complete separator in $G$. The removal of any link $V_i V_j \in \mathcal{S}$ disconnects $T$ into two subtrees which we denote $T^{(i)}$ and $T^{(j)}$ (chosen so that $T^{(i)}$ contains $V_i$). We will let $\mathcal{K}^{(i)}$ be the nodes of $T^{(i)}$, and $V^{(i)} = \cup_{V \in K^{(i)}} V$ be the set of vertices corresponding to the subtree $T^{(i)}$. The junction tree property ensures that $V^{(i)} \cap V^{(j)} = V_i \cap V_j = V_{ij}$. We will let $G^{(i)}$ be the subgraph induced by $V^{(i)}$.

A graphical model is a pair $(G, P)$ where $P$ is the joint probability distribution for random variables $X_1, X_2, \ldots, X_n$, and $G$ is a graph with vertex set $V(G) = \{X_1, X_2, \ldots, X_n\}$ such that the separators in $G$ imply conditional independencies in $P$ (so $P$ *factors* according to $G$). If $G$ is triangulated, then the junction-tree algorithm can be used for exact inference in the probability distribution $P$. The complexity of this algorithm grows with the treewidth of $G$ (which is one less than the size of the largest clique in $G$ when $G$ is triangulated). The growth is exponential when $P$ is a discrete probability distribution, thus rendering exact inference for graphs with large treewidth impractical. Therefore, we seek another graphical model $(H, P_H)$ which allows tractable inference (so $H$ should have lower treewidth than $G$ has). The general problem of finding a graphical model of tree-width at most $k$ so as to minimize the KL-divergence from a specified probability distribution is NP complete for general $k$ ([9]) However, it is known that this problem is solvable in polynomial time (in $|V(G)|$) for some special cases cases (such as when $G$ has bounded treewidth or when $k = 1$ [1]). If $(G, P_G)$ and $(H, P_H)$ are graphical models, then we say that $(H, P_H)$ is a subgraphical model of $(G, P_G)$ if $H$ is a spanning subgraph of $G$. Note in particular that separators in $G$ are separators in $H$, and hence $(G, P_H)$ is also a graphical model.

## 2 Graph Decompositions and Divide-and-Conquer Algorithms

For the remainder of the paper, we will be assuming that $G = (V, E)$ is some triangulated graph, with junction tree $T = (\mathcal{K}, \mathcal{S})$. As observed above, if $V_i V_j \in \mathcal{S}$, then the removal

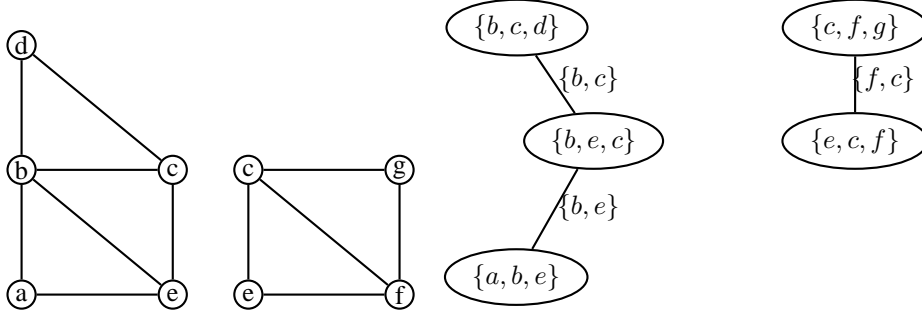

Figure 2: The graphs $G^{(i)}, G^{(j)}$ and junction-trees $T^{(i)}$ and $T^{(j)}$ resulting from the removal of the link $V_{ij} = \{c, e\}$

of $V_{ij} = V_i \cap V_j$ disconnects $G$ into two (vertex-induced) subgraphs $G^{(i)}$ and $G^{(j)}$ which are both triangulated, with junction-trees $T^{(i)}$ and $T^{(j)}$ respectively. We can recursively decompose each of $G^{(i)}$ and $G^{(j)}$ into smaller and smaller subgraphs till the resulting subgraphs are cliques. When the size of all the minimal separators are bounded, we may use these decompositions to easily solve problems that are hard in general. For example, in [5] it is shown that NP-complete problems like vertex coloring, and finding maximum independent sets can be solved in polynomial time on graphs with bounded tree-width (which are equivalent to spanning graphs with bounded size separators). We will be interested in finding (triangulated) subgraphs of $G$ that satisfy some conditions, such as a bound on the number of edges, or a bound on the tree-width and which optimize *separable* objective functions (described in Section 2)

One reason why problems such as this can often be solved easily when the tree-width of $G$ is bounded by some constant is this : If $V_{ij}$ is a separator decomposing $G$ into $G^{(i)}$ and $G^{(j)}$, then a divide-and-conquer approach would suggest that we try and find optimal subgraphs of $G^{(i)}$ and $G^{(j)}$ and then *splice* the two together to get an optimal subgraph of $G$. There are two issues with this approach. First, the optimal subgraphs of $G^{(i)}$ and $G^{(j)}$ need not necessarily match up on $V_{ij}$, the set of common vertices. Second, even if the two subgraphs agree on the set of common vertices, the graph resulting from splicing the two subgraphs together need not be triangulated (which could happen even if the two subgraphs individually are triangulated). To rectify the situation, we can do the following. We partition the set of subgraphs of $G^{(i)}$ and $G^{(j)}$ into classes, so that any subgraph of $G^{(i)}$ and any subgraph $G^{(j)}$ corresponding to the same class are compatible in the sense that they match up on their intersection namely $V_{ij}$, and so that by splicing the two subgraphs together, we get a subgraph of $G$ which is acceptable (and in particular is triangulated). Then given optimal subgraphs of both $G^{(i)}$ and $G^{(j)}$ corresponding to each class, we can enumerate over all the classes and pick the best one. Of course, to ensure that we do not repeatedly solve the same problem, we need to work bottom-up (a.k.a dynamic programming) or memoize our solutions. This procedure can be carried out in polynomial (in $|V|$) time as long as we have only a polynomial number of classes. Now, if we have a polynomial number of classes, these classes need not actually be a partition of all the acceptable subgraphs, though the union of the classes must cover all acceptable subgraphs (so the same subgraph can be contained in more than one class). For our application, every class can be thought of to be the set of subgraphs that satisfy some constraint, and we need to pick a polynomial number of constraints that cover all possibilities. The bound on the tree-width helps us here. If $|V_{ij}| = k$, then in any subgraph $H$ of $G$, $H[V_{ij}]$ must be one of the $2^{\binom{k}{2}}$ possible subgraphs of $G[V_{ij}]$. So, if $k$ is sufficiently small (so $2^{\binom{k}{2}}$ is bounded by some polynomial in $|V|$),

then this procedure results in a polynomial time algorithm. In this paper, we show that in some cases we can characterize the space $\mathcal{H}$ so that we still have a polynomial number of constraints even when the tree-width of $G$ is not bounded by a small constant.

## 2.1 Separable objective functions

For cases where exact inference in the graphical model $(G, P_G)$ is intractable, it is natural to try to find a subgraphical model $(H, P_H)$ such that $D(P_G\|P_H)$ is minimized, and inference using $H$ is tractable. We will denote by $\mathcal{H}$ the set of subgraphs of $G$ that are tractable for inference. For example, this set could be the set of subgraphs of $G$ with treewidth one less than the treewidth of $G$, or perhaps the set of subgraphs of $G$ with at $d$ fewer edges. For a specified subgraph $H$ of $G$, there is a unique probability distribution $P_H$ factoring over $H$ that minimizes $D(P_G\|P_H)$. Hence, finding a optimal subgraphical model is equivalent to finding a subgraph $H$ for which $D(P_G\|P_H)$ is minimized. If $V_{ij}$ is a separator of $G$, we will attempt to find optimal subgraphs of $G$ by finding optimal subgraphs of $G^{(i)}$ and $G^{(j)}$ and splicing them together. However, to do this, we need to ensure that the objective criteria also decomposes along the separator $V_{ij}$. Suppose that $H$ is any triangulated subgraph of $G$. Let $P_{G^{(i)}}$ and $P_{G^{(j)}}$ be the (marginalized) distributions of $P_G$ on $V^{(i)}$ and $V^{(j)}$ respectively, and $P_{H^{(i)}}$ and $P_{H^{(j)}}$ be the (marginalized) distributions of the distribution $P_H$ on $V^{(i)}$ and $V^{(j)}$ where $H^{(i)} = H[V^{(i)}]$ and $H^{(j)} = H[V^{(j)}]$, The following result assures us that the KL-divergence also factors according to the separator $V_{ij}$.

**Lemma 1.** *Suppose that $(G, P_G)$ is a graphical model, $H$ is a triangulated subgraph of G, and $P_H$ factors over $H$. Then $D(P_G\|P_H) = D(P_{G^{(i)}}\|P_{H^{(i)}}) + D(P_{G^{(j)}}\|P_{H^{(j)}}) - D(P_{G[V_{ij}]}\|P_{H[V_{ij}]})$.*

*Proof.* Since $H$ is a subgraph of $G$, and $V_{ij}$ is a separator of $G$, $V_{ij}$ must also be a separator of $H$. Therefore, $P_H\left(\{X_v\}_{v\in V}\right) = \frac{P_{H^{(i)}}(\{X_v\}_{v\in V^{(i)}})\cdot P_{H^{(j)}}(\{X_v\}_{v\in V^{(j)}})}{P_{H[V_{ij}]}(\{X_v\}_{v\in V_{ij}})}$. The result follows immediately. $\qquad\square$

Therefore, there is hope that we can reduce our our original problem of finding an optimal subgraph $H \in \mathcal{H}$ as one of finding subgraphs of $H^{(i)} \subseteq G^{(i)}$ and $H^{(j)} \subseteq G^{(j)}$ that are compatible, in the sense that they match up on the overlap $V_{ij}$, and for which $D(P_G\|P_H)$ is minimized. Throughout this paper, for the sake of concreteness, we will assume that the objective criterion is to minimize the KL-divergence. However, all the results can be extended to other objective functions, as long as they "separate" in the sense that for any separator, the objective function is the sum of the objective functions of the two parts, possibly modulo some correction factor which is purely a function of the separator. Another example might be the complexity $r(H)$ of representing the graphical model $H$. A very natural representation satisfies $r(G) = r(G^{(i)}) + r(G^{(j)})$ if $G$ has a separator $G^{(i)} \cap G^{(j)}$. Therefore, the representation cost reduction would satisfy $r(G) - r(H) = (r(G^{(i)}) - r(H^{(i)})) + (r(G^{(j)}) - r(H^{(j)}))$, and so also factors according to the separators. Finally note that any linear combinations of such separable functions is also separable, and so this technique could also be used to determine tradeoffs (representation cost vs. KL-divergence loss for example). In Section 4 we discuss some issues regarding computing this function.

## 2.2 Decompositions and decomposition trees

For the algorithms considered in this paper, we will be mostly interested in the decompositions that are specified by the junction tree, and we will represent these decompositions by a rooted tree called a *decomposition tree*. This representation was introduced in [2, 3], and is similar in spirit to Darwiche's dtrees [6] which specify decompositions of directed acyclic graphs. In this section and the next, we show how a decomposition tree for a graph may be constructed, and show how it is used to solve a number of optimization problems.

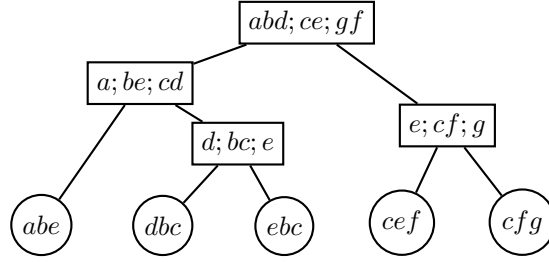

Figure 3: The separator tree corresponding to Figure 1

A *decomposition tree* for $G$ is a rooted tree whose vertices correspond to separators and cliques of $G$. We describe the construction of the decomposition tree in terms of a junction-tree $T = (\mathcal{K}, \mathcal{S})$ for $G$. The interior nodes of the decomposition tree $R(T)$ correspond to $\mathcal{S}$ (the links of $T$ and hence the minimal separators of $G$). The leaf or terminal nodes represent the elements of $\mathcal{K}$ (the nodes of $T$ and hence the maximal cliques of $G$). $R(T)$ can be recursively constructed from $T$ as follows : If $T$ consists of just one node $K$, (and hence no edges), then $R$ consists of just one node, which is given the label $K$ as well. If however, $T$ has more than one node, then $T$ must contain at least one link. To begin, let $V_i V_j \in \mathcal{S}$ be any link in $T$. Then removal of the link $V_i V_j$ results in two disjoint junction-trees $T^{(i)}$ and $T^{(j)}$. We label the root of $R$ by the decomposition $(V^{(i)}; V_{ij}; V^{(j)})$. The rest of $R$ is recursively built by successively picking links of $T^{(i)}$ and $T^{(j)}$ (decompositions of $G^{(i)}$ and $G^{(j)}$) to form the interior nodes of $R$. The effect of this procedure on the junction tree of Figure 1 is shown in Figure 3, where the decomposition associated with the interior nodes is shown inside the nodes. Let $\mathcal{M}$ be the set of all nodes of $R(T)$. For any interior node $M$ induced by the the link $V_i V_j \in \mathcal{S}$ of $T$, then we will let $M^{(i)}$ and $M^{(j)}$ represent the left and right children of $M$, and $R^{(i)}$ and $R^{(j)}$ be the left and right trees below $M$.

## 3    Finding optimal subgraphical models

### 3.1    Optimal sub $(k-1)$-trees of $k$-trees

Suppose that $G$ is a $k$-tree. A sub $(k-1)$-tree of $G$ is a subgraph $H$ of $G$ that is $(k-1)$-tree. Now, if $V_{ij}$ is any minimal separator of $G$, then both $G^{(i)}$ and $G^{(j)}$ are $k$-trees on vertex sets $V^{(i)}$ and $V^{(j)}$ respectively. It is clear that the induced subgraphs $H[V^{(i)}]$ and $H[V^{(j)}]$ are subgraphs of $G^{(i)}$ and $G^{(j)}$ and are **partial** $(k-1)$-trees. We will be interested in finding sub $(k-1)$-trees of $k$ trees and this problem is trivial by the result of [1] when $k = 2$. Therefore, we assume that $k \geq 3$. The following result characterizes the various possibilities for $H[V_{ij}]$ in this case.

**Lemma 2.** *Suppose that $G$ is a $k$-tree, and $S = V_{ij}$ is a minimal separator of $G$ corresponding to the link $ij$ of the junction-tree $T$. In any $(k-1)$-tree $H \subseteq G$ either*

1. *There is a $u \in S$ such that $u$ is not connected to vertices in both $V^{(i)} \setminus S$ and $V^{(j)} \setminus S$. Then $S \setminus \{u\}$ is a minimal separator in $H$ and hence is complete.*

2. *Every vertex in $S$ is connected to vertices in both $V^{(i)} \setminus S$ and $V^{(j)} \setminus S$. Then there are vertices $\{x, y\} \subseteq S$ such that the edge $H[S]$ is missing only the edge $\{x, y\}$. Further either $H[V^{(i)}]$ or $H[V^{(j)}]$ does not contain a unchorded $x$-$y$ path.*

*Proof.* We consider two possibilities. In the first, there is some vertex $u \in S$ such that $u$ is not connected to vertices in both $V^{(i)} \setminus S$ and $V^{(j)} \setminus$. Since the removal of $S$ disconnects $G$, the removal of $S$ must also disconnect $H$. Therefore, $S$ must contain a minimal separator of $H$. Since $H$ is a $(k-1)$-tree, all minimal separators of $H$ must contain $k-1$ vertices which must therefore be $S \setminus \{u\}$. This corresponds to case (1) above. Clearly this possiblity can occur.

If there is no such $u \in S$, then every vertex in $S$ is connected to vertices in both $V^{(i)} \setminus S$ and $V^{(j)} \setminus S$. If $x \in S$ is connected to some $y_i \in V^{(i)} \setminus S$ and $y_j \in V^{(j)} \setminus S$, then $x$ is contained in every minimal $y_i/y_j$ separator (see [5]). Therefore, every vertex in $S$ is part of a minimal separator. Since each minimal separator contains $k-1$ vertices, there must be at least two distinct minimum separators contained in $S$. Let $S_x = S \setminus \{x\}$ and $S_y = S \setminus \{y\}$ be two distinct minimal separators. We claim that $H[S]$ contains all edges except the edge $\{x, y\}$. To see this, note that if $z, w \in S$, with $z \neq w$ and $\{z, w\} \neq \{x, y\}$ (as sets), then either $\{z, w\} \subset S_y$ or $\{z, w\} \subset S_x$. Since both $S_x$ and $S_y$ are complete in $H$, this edge must be present in $H$. The edge $\{x, y\}$ is not present in $H[S]$ because all minimal separators in $H$ must be of size $k-1$. Further, if both $V^{(i)}$ and $V^{(j)}$ contain an unchorded path between $x$ and $y$, then by joining the two paths at $x$ and $y$, we get a unchorded cycle in $H$ which contradicts the fact that $H$ is triangulated. $\square$

Therefore, we may associate $\binom{k}{2} \cdot 2 + 2 \cdot k$ constraints with each separator $V_{ij}$ of $G$ as follows. There are $k$ possible constraints corresponding to case (1) above (one for each choice of $x$), and $\binom{k}{2} \cdot 2$ choices corresponding to case (2) above. This is because for each pair $\{x, y\}$ corresponding to the missing edge, we have either $V^{(i)}$ contains no unchorded $xy$ paths or $V^{(j)}$ contains no unchorded $xy$ paths. More explicitly, we can encode the set of constraints $\mathcal{C}_M$ associated with each separator $S$ corresponding to an interior node $M$ of the decomposition tree as follows: $\mathcal{C}_M = \{(x, y, s) : x \in S, y \in S, s \in \{i, j\}\}$. If $y = x$, then this corresponds to case (1) of the above lemma. If $s = i$, then $x$ is connected only to $H^{(i)}$ and if $s = j$, then $x$ is connected only to $H^{(j)}$. If $y \neq x$, then this corresponds to case (2) in the above lemma. If $s = i$, then $H^{(i)}$ does not contain any unchorded path between $x$ and $y$, and there is no constraint on $H^{(j)}$. Similarly if $s = j$, then $H^{(j)}$ does not contain any unchorded path between $x$ and $y$, and there is no constraint on $H^{(i)}$.

Now suppose that $H^{(i)}$ and $H^{(j)}$ are triangulated subgraphs of $G^{(i)}$ and $G^{(j)}$ respectively, then it is clear that if $H^{(i)}$ and $H^{(j)}$ both satisfy the same constraint they must match up on the common vertices $V_{ij}$. Therefore to splice together two solutions corresponding to the same constraint, we only need to check that the graph obtained by splicing the graphs is triangulated.

**Lemma 3.** *Suppose that $H^{(i)}$ and $H^{(j)}$ are triangulated subgraphs of $G^{(i)}$ and $G^{(j)}$ respectively such that both of them satisfy the same constraint as described above. Then the graph $H$ obtained by splicing $H^{(i)}$ and $H^{(j)}$ together is triangulated.*

*Proof.* Suppose that both $H^{(i)}$ and $H^{(j)}$ are both triangulated and both satisfy the same constraint. If both $H^{(i)}$ and $H^{(j)}$ satisfy the same constraint corresponding to case (1) in Lemma 2 and $H$ has an unchorded cycle, then this cycle must involve elements of both $H^{(i)}$ and $H^{(j)}$. Therefore, there must be two vertices of $S \setminus \{u\}$ on the cycle, and hence this cycle has a chord as $S \setminus \{u\}$ is complete. This contradiction shows that $H$ is triangulated. So assume that both of them satisfy the constraint corresponding to case (2) of Lemma 2. Then if $H$ is not triangulated, there must be a $t$-cycle (for $t \geq 4$) with no chord. Now, since $\{x, y\}$ is the only missing edge of $S$ in $H$, and because $H^{(i)}$ and $H^{(j)}$ are individually triangulated, the cycle must contain $x$, $y$ and vertices of both $V^{(i)} \setminus S$ and $V^{(j)} \setminus S$. We

may split this unchorded cycle into two unchorded paths, one contained in $V^{(i)}$ and one in $V^{(j)}$ thus violating our assumption that both $H^{(i)}$ and $H^{(j)}$ satisfy the same constraint. □

If $|S| = k$, then there are $2k + 2 \cdot \binom{k}{2} \in O(k^2) \in O(n^2)$. We can use a divide and conquer strategy to find the optimal sub $(k-1)$ tree once we have taken care of the base case, where $G$ is just a single clique (of $k+1$ elements). However, for this case, it is easily checked that any subgraph of $G$ obtained by deleting exactly one edge results in a $(k-1)$ tree, and every sub $(k-1)$-tree results from this operation. Therefore, the optimal $(k-1)$-tree can be found using Algorithm 1, and in this case, the complexity of Algorithm 1 is $O(n(k+1)^2)$. This procedure can be generalized to find the optimal sub $(k-d)$- tree for any fixed $d$. However, the number of constraints grows exponentially with $d$ (though it is still polynomial in $n$). Therefore for small, fixed values of $d$, we can compute the optimal sub $(k-d)$-tree of $G$. While we can compute $(k-d)$-trees of $G$ by first going from a $k$ tree to a $(k-1)$ tree, then from a $(k-1)$-tree to a $(k-2)$-tree, and so on in a greedy fashion, this will not be optimal in general. However, this might be a good algorithm to try when $d$ is large.

## 3.2 Optimal triangulated subgraphs with $|E(G)| - d$ edges

Suppose that we are interested in a (triangulated) subgraph of $G$ that contains $d$ fewer edges that $G$ does. That is, we want to find an optimal subgraph $H \subset G$ such that $|E(H)| = |E(G)| - d$. Note that by the result of [4] there is always a triangulated subgraph with $d$ fewer edges (if $d < |E(G)|$). Two possibilities for finding such an optimal subgraph are

1. Use the procedure described in [4]. This is a greedy procedure which works in $d$ steps by deleting an edge at each step. At each state, the edge is picked from the set of edges whose deletion leaves a triangulated graph. Then the edge which causes the least increase in KL-divergence is picked at each stage.

2. For each possible subset $A$ of $E(G)$ of size $d$, whose deletion leaves a triangulated graph, compute the KL divergence using the formula above, and then pick the optimal one. Since there are $\binom{|E(G)|}{d}$ such sets, this can be done in polynomial time (in $|V(G)|$) when $d$ is a constant.

The first greedy algorithm is not guaranteed to yield the optimal solution. The second takes time that is $O(n^{2d})$. Now, let us solve this problem using the framework we've described.

Let $\mathcal{H}$ be the set of subgraphs of $G$ which may be obtained by deletion of $d$ edges. For each $M = ij \in \mathcal{M}$ corresponding to the separator $V_{ij}$, let $\mathcal{C}_M = \left\{ (l, r, c, s, A) : l + r - c = d, s \text{ a } d \text{ bit string}, A \in \binom{E(G[V_{ij}])}{c} \right\}$. The constraint represented by $(l, r, c, A)$ is this : $A$ is a set of $d$ edges of $G[V_{ij}]$ that are missing in $H$, $l$ edges are missing from the left subgraph, and $r$ edges are missing from the right subgraph. $c$ represents the double count, and so is subtracted from the total. If $k$ is the size of the largest clique, then the total number of such constraints is bounded by $2d \cdot 2^d \cdot \binom{\binom{k}{2}}{d} \in O(k^{2d})$ which could be better than $O(n^{2d})$ and is polynomial in $|V|$ when $d$ is constant. See [10] for additional details.

## 4 Conclusions

Algorithm 1 will compute the optimal $H \in \mathcal{H}$ for the two examples discussed above and is polynomial (for fixed constant $d$) even if $k$ is $O(n)$. In [10] a generalization is presented which will allow finding the optimal solution for other classes of subgraphical models. Now, we assume an oracle model for computing KL-divergences of probability distributions on vertex sets of cliques. It is clear that these KL-divergences can be computed

$R \leftarrow$ separator-tree for $G$;

**for** *each vertex $M$ of $R$ in order of increasing height (bottom up)* **do**

    **for** *each constraint $c_M$ of $M$* **do**

        **if** $M$ *is an interior vertex of $R$ corresponding to edge $ij$ of the junction tree* **then**

            Let $M_l$ and $M_r$ be the left and right children of $M$;

            Pick constraint $c_l \in \mathcal{C}_{M_l}$ compatible with $c_M$ to minimize $\mathsf{table}[M_l, c_l]$;

            Pick constraint $c_r \in \mathcal{C}_{M_r}$ compatible with $c_M$ to minimize $\mathsf{table}[M_r, c_r]$;

            $\mathsf{loss} \leftarrow D(P_G[M] \| P_H[M])$;

            $\mathsf{table}[M, c_M] \leftarrow \mathsf{table}[M_l, c_l] + \mathsf{table}[M_r, c_r] - \mathsf{loss}$;

        **else**

            $\mathsf{table}[M, c_M] \leftarrow D(P_G[M] \| P_H[M])$;

        **end**

    **end**

**end**

**Algorithm 1: Finding optimal set of constraints**

efficiently for distributions like Gaussians, but for discrete distributions this may not be possible when $k$ is large. However even in this case this algorithm will result in only polynomial calls to the oracle. The standard algorithm [3] which is exponential in the treewidth will make $O(2^k)$ calls to this oracle. Therefore, when the cost of computing the KL-divergence is large, this algorithm becomes even more attractive as it results in exponential speedup over the standard algorithm. Alternatively, if we can compute approximate KL-divergences, or approximately optimal solutions, then we can compute an approximate solution by using the same algorithm.

## Footnotes

\*This work was supported by NSF grant IIS-0093430 and an Intel Corporation Grant.

## References

[1] C. Chow and C. Liu, "Approximating discrete probability distributions with dependence trees", IEEE Transactions on Information Theory, v. 14, 1968, Pages 462–467.

[2] F. Gavril, "Algorithms on clique separable graphs", Discrete Mathematics v. 9 (1977), pp. 159–165.

[3] R. E. Tarjan. "Decomposition by Clique Separators", Discrete Mathematics, v. 55 (1985), pp. 221–232.

[4] U. Kjaerulff. "Reduction of computational complexity in Bayesian networks through removal of weak dependencies", Proceedings of the Tenth Annual Conference on Uncertainty in Artificial Intelligence, pp. 374–382, 1994.

[5] T. Kloks, "Treewidth: Computations and Approximations", Springer-Verlag, 1994.

[6] A. Darwiche and M. Hopkins. "Using recursive decomposition to construct elimination orders, jointrees and dtrees", Technical Report D-122, Computer Science Dept., UCLA.

[7] S. Lauritzen. "Graphical Models", Oxford University Press, Oxford, 1996.

[8] T. A. McKee and F. R. McMorris. "Topics in Intersection Graph Theory", SIAM Monographs on Discrete Mathematics and Applications, 1999.

[9] D. Karger and N. Srebro. "Learning Markov networks: Maximum bounded tree-width graphs." In *Symposium on Discrete Algorithms*, 2001, Pages 391-401.

[10] M. Narasimhan and J. Bilmes. "Optimization on separator-clique trees.", Technical report UWEETR 2004-10, June 2004.
